# Family Discovery

**Stephen M. Omohundro**
NEC Research Institute
4 Independence Way, Princeton, NJ 08540
om@research.nj.nec.com

## Abstract

"Family discovery" is the task of learning the dimension and structure of a parameterized family of stochastic models. It is especially appropriate when the training examples are partitioned into "episodes" of samples drawn from a single parameter value. We present three family discovery algorithms based on surface learning and show that they significantly improve performance over two alternatives on a parameterized classification task.

## 1   INTRODUCTION

Human listeners improve their ability to recognize speech by identifying the accent of the speaker. "Might" in an American accent is similar to "mate" in an Australian accent. By first identifying the accent, discrimination between these two words is improved. We can imagine locating a speaker in a "space of accents" parameterized by features like pitch, vowel formants, "r"-strength, etc. This paper considers the task of learning such parameterized models from data.

Most speech recognition systems train hidden Markov models on labelled speech data. Speaker-dependent systems train on speech from a single speaker. Speaker-independent systems are usually similar, but are trained on speech from many different speakers in the hope that they will then recognize them all. This kind of training ignores speaker identity and is likely to result in confusion between pairs of words which are given the same pronunciation by speakers with different accents.

Speaker-independent recognition systems could more closely mimic the human approach by using a learning paradigm we call "family discovery". The system would be trained on speech data partitioned into "episodes" for each speaker. From this data, the system would construct a *parameterized family* of models representing dif-

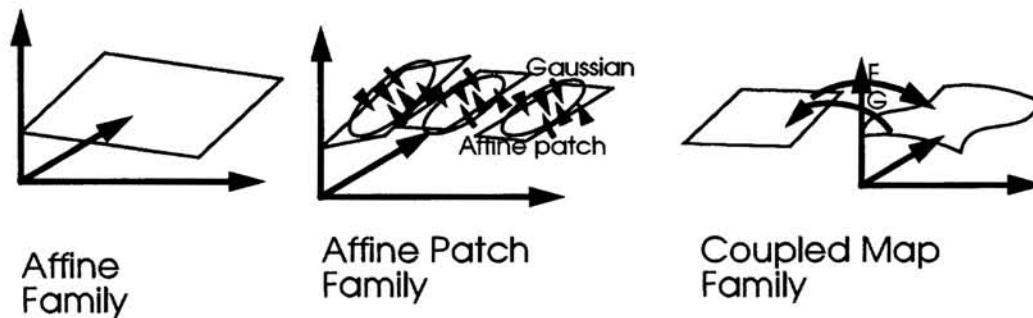

Figure 1: The structure of the three family discovery algorithms.

ferent accents. The learning algorithms presented in this paper could determine the dimension and structure of the parameterization. Given a sample of new speech, the best-fitting accent model would be used for recognition.

The same paradigm applies to many other recognition tasks. For example, an OCR system could learn a parameterized family of font models (Revow, et. al., 1994). Given new text, the system would identify the document's font parameters and use the corresponding character recognizer.

In general, we use "family discovery" to refer to the task of learning the dimension and structure of a parameterized family of stochastic models. The methods we present are equally applicable to parameterized density estimation, classification, regression, manifold learning, reinforcement learning, clustering, stochastic grammar learning, and other stochastic settings. Here we only discuss classification and primarily consider training examples which are explicitly partitioned into episodes.

This approach fits naturally into the neural network literature on "meta-learning" (Schmidhuber, 1995) and "network transfer" (Pratt, 1994). It may also be considered as a particular case of the "bias learning" framework proposed by Baxter at this conference (Baxter, 1996).

There are two primary alternatives to family discovery: 1) try to fit a single model to the data from all episodes or 2) use separate models for each episode. The first approach ignores the information that the different training sets came from distinct models. The second approach eliminates the possibility of inductive generalization from one set to another.

In Section 2, we present three algorithms for family discovery based on techniques for "surface learning" (Bregler and Omohundro, 1994 and 1995). As shown in Figure 1, the three alternative representations of the family are: 1) a single affine subspace of the parameter space, 2) a set of local affine patches smoothly blended together, and 3) a pair of coupled maps from the parameter space into the model space and back. In Section 3, we compare these three approaches to the two alternatives on a parameterized classification task.

## 2    THE FIVE ALGORITHMS

Let the space of all classifiers under consideration be parameterized by $\theta$ and assume that different values of $\theta$ correspond to different classifiers (ie. it is identifiable). For example, $\theta$ might represent the means, covariances, and class priors of a classifier with normal class-conditional densities. $\theta$-space will typically have a much higher dimension than the parameterized family we are seeking. We write $p_\theta(x)$ for the total probability that the classifier $\theta$ assigns to a labelled or unlabelled example $x$.

The true models are drawn from a $d$-dimensional family parameterized by $\gamma$. Let the training set be partitioned into $N$ episodes where episode $i$ consists of $N_i$ training examples $t_{ij}$, $1 \leq j \leq N_i$ drawn from a single underlying model with parameter $\theta_i^*$. A family discovery learning algorithm uses this training data to estimate the underlying parameterized family.

From a parameterized family, we may define the projection operator $P$ from $\theta$-space to itself which takes each $\theta$ to the closest member of the family. Using this projection operator, we may define a "family prior" on $\theta$-space which dies off exponentially with the square distance of a model from the family $m_P(\theta) \propto e^{-(\theta - P(\theta))^2}$. Each of the family discovery algorithms chooses a family so as to maximize the posterior probability of the training data with respect to this prior. If the data is very sparse, this MAP approximation to a full Bayesian solution can be supplemented by "Occam" terms (MacKay, 1995) or by using a Monte Carlo approximation.

The outer loop of each of the algorithms performs the optimization of the fit of the data by re-estimation in a manner similar to the Expectation Maximization (EM) approach (Jordan and Jacobs, 1994). First, the training data in each episode $i$ is independently fit by a model $\theta_i$. Then the dimension of the family is determined as described later and the family projection operator $P$ is chosen to maximize the probability that the episode models $\theta_i$ came from that family $\prod_i m_P(\theta_i)$. The episode models $\theta_i$ are then re-estimated including the new prior probability $m_P$. These newly re-estimated models are influenced by the other episodes through $m_P$ and so exhibit training set "transfer". The re-estimation loop is repeated until nothing changes.

The learned family can then be used to classify a set of $N_{test}$ unlabelled test examples $x_k$, $1 \leq k \leq N_{test}$ drawn from a model $\theta_{test}^*$ in the family. First, the parameter $\theta_{test}$ is estimated by selecting the member of the family with the highest likelihood on the test samples. This model is then used to perform the classification. A good approximation to the best-fit family member is often to take the image of the best-fit model in the entire $\theta$-space under the projection operator $P$.

In the next five sections, we describe the two alternative approaches and the three family discovery algorithms. They differ only in their choice of family representation as encoded in the projection operator $P$.

### 2.1    The Single Model Approach

The first alternative approach is to train a single model on all of the training data. It selects $\theta$ to maximize the total likelihood $L(\theta) = \prod_{i=1}^{N} \prod_{j=1}^{N_i} p_\theta(t_{ij})$. New test data is classified by this single selected model.

## 2.2  The Separate Models Approach

The second alternative approach fits separate models for each training episode. It chooses $\theta_i$ for $1 \leq i \leq N$ to maximize the episode likelihood $L_i(\theta_i) = \prod_{j=1}^{N_i} p_\theta(t_{ij})$. Given new test data, it determines which of the individual models $\theta_i$ fit best and classifies the data with it.

## 2.3  The Affine Algorithm

The affine model represents the underlying model family as an affine subspace of the model parameter space. The projection operator $P_{affine}$ projects a parameter vector $\theta$ orthogonally onto the affine subspace. The subspace is determined by selecting the top principal vectors in a principal components analysis of the best-fit episode model parameters. As described in (Bregler & Omohundro, 1994) the dimension is chosen by looking for a gap in the principal values.

## 2.4  The Affine Patch Algorithm

The second family discovery algorithm is based on the "surface learning" procedure described in (Bregler and Omohundro, 1994). The family is represented by a collection of local affine patches which are blended together using Gaussian influence functions. The projection mapping $P_{patch}$ is a smooth convex combination of projections onto the affine patches $P_{patch}(\theta) = \sum_{\alpha=1}^{m} I_\alpha(\theta)A_\alpha(\theta)$ where $A_\alpha$ is the projection operator for an affine patch and $I_\alpha(\theta) = \frac{G_\alpha(\theta)}{\sum_\alpha G_\alpha(\theta)}$ is a normalized Gaussian blending function.

The patches are initialized using $k$-means clustering on the episode models to choose $k$ patch centers. A local principal components analysis is performed on the episode models which are closest to each center. The family dimension is determined by examining how the principal values scale as successive nearest neighbors are considered. Each patch may be thought of as a "pancake" lying in the surface. Dimensions which belong to the surface grow quickly as more neighbors are considered while dimensions across the surface grow only because of the curvature of the surface.

The Gaussian influence functions and the affine patches are then updated by the EM algorithm (Jordan and Jacobs, 1994). With the affine patches held fixed, the Gaussians $G_\alpha$ are refit to the errors each patch makes in approximating the episode models. Then with the Gaussians held fixed, the affine patches $A_\alpha$ are refit to the epsiode models weighted by the the corresponding Gaussian $G_\alpha$. Similar patches may be merged together to form a more parsimonious model.

## 2.5  The Coupled Map Algorithm

The affine patch approach has the virtue that it can represent topologically complex families (eg. families representing physical objects might naturally be parameterized by the rotation group which is topologically a projective plane). It cannot, however, provide an explicit parameterization of the family which is useful in some applications (eg. optimization searches). The third family discovery algorithm therefore attempts to directly learn a parameterization of the model family.

Recall that the model parameters define $\theta$-space, while the family parameters de-

fine $\gamma$-space. We represent a family by a mapping $G$ from $\theta$-space to $\gamma$-space together with a mapping $F$ from $\gamma$-space back to $\theta$-space. The projection operation is $P_{map}(\theta) = F(G(\theta))$. The map $G(\theta)$ defines the family parameter $\gamma$ on the full $\theta$-space.

This representation is similar to an "auto-associator" network in which we attempt to "encode" the best-fit episode parameters $\theta_i$ in the lower dimensional $\gamma$-space by the mapping $G$ in such a way that they can be correctly reconstructed by the function $F$. Unfortunately, if we try to train $F$ and $G$ using back-propagation on the identity error function, we get no training data away from the family. There is no reason for $G$ to project points away from the family to the closest family member. We can rectify this by training $F$ and $G$ iteratively. First an arbitrary $G$ is chosen and $F$ is trained to send the images $\gamma_i = G(\theta_i)$ back to $\theta_i$. $G$ is trained, however, on images under $F$ corrupted by additive spherical Gaussian noise! This provides samples away from the family and on average the training signal sends each point in $\theta$ space to the closest family member.

To avoid iterative training, our experiments used a simpler approach. $G$ was taken to be the affine projection operator defined by a global principal components analysis of the best-fit episode model parameters. Once $G$ is defined, $F$ is chosen to minimize the difference between $F(G(\theta_i))$ and $\theta_i$ for each best-fit episode parameter $\theta_i$.

Any form of trainable nonlinear mapping could be used for $F$ (eg. backprop neural networks or radial basis function networks). We represent $F$ as a mixture of experts (Jordan and Jacobs, 1994) where each expert is an affine mapping and the mixture coefficients are Gaussians. The mapping is trained by the EM algorithm.

# 3    ALGORITHM COMPARISON

To compare these five algorithms, we consider a two-class classification task with unit-variance normal class-conditional distributions on a 5-dimensional feature space. The means of the class distributions are parameterized by a nonlinear two-parameter family:

$$\begin{aligned} m_1 &= (\gamma_1 + \tfrac{1}{2}\cos\phi)\hat{e}_1 + (\gamma_2 + \tfrac{1}{2}\sin\phi)\hat{e}_2 \\ m_2 &= (\gamma_1 - \tfrac{1}{2}\cos\phi)\hat{e}_1 + (\gamma_2 - \tfrac{1}{2}\sin\phi)\hat{e}_2. \end{aligned}$$

where $0 \le \gamma_1, \gamma_2 \le 10$ and $\phi = (\gamma_1 + \gamma_2)/3$. The class means are kept at a unit distance apart, ensuring significant class overlap over the whole family. The angle $\phi$ varies with the parameters so that the correct classification boundary changes orientation over the family. This choice of parameters introduces sufficient nonlinearity in the task to distinguish the non-linear algorithms from the linear one.

Figure 1 shows the comparative performance of the 5 algorithms. The $x$-axis is the total number of training examples. Each set of examples consisted of approximately $N = \sqrt{x}$ episodes of approximately $N_i = \sqrt{x}$ examples each. The classifier parameters for an episode were drawn uniformly from the classifier family. The episode training examples were then sampled from the chosen classifier according to the classifier's distribution. Each of the 5 algorithms was then trained on these examples. The number of patches in the surface patch algorithm and the number of affine components in the surface map algorithm were both taken to be the square-root of

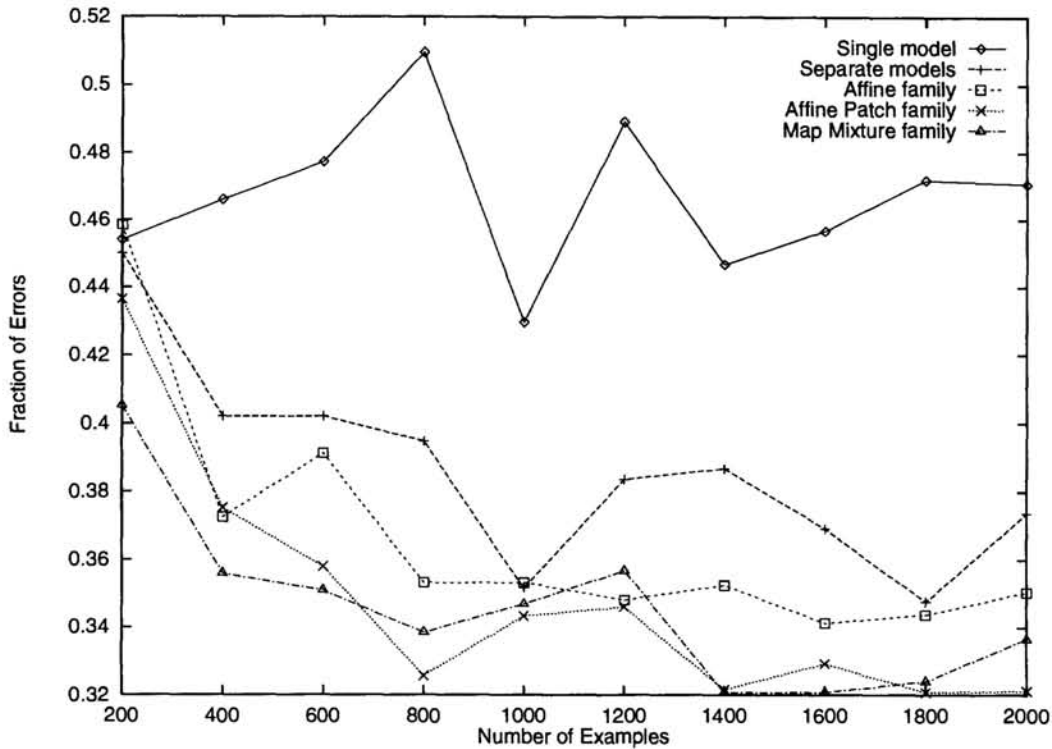

Figure 2: A comparison of the 5 family discovery algorithms on the classification task.

the number of training episodes.

The $y$-axis shows the percentage correct for each algorithm on an independent test set. Each test set consisted of 50 episodes of 50 examples each. The algorithms were presented with unlabelled data and their classification predictions were then compared with the correct classification label.

The results show significant improvement through the use of family discovery for this classification task. The single model approach performed significantly worse than any of the other approaches, especially for larger numbers of episodes (where the family discovery becomes possible). The separate model approach improves with the number of episodes, but is nearly always bested by the approaches which take explicit account of the underlying parameterized family. Because of the nonlinearity in this task, the simple affine model performs more poorly than the two nonlinear methods. It is simple to implement, however, and may well be the method of choice when the parameters aren't so nonlinear. From this data, there is not a clear winner between the surface patch and surface map approaches.

## 4   TRAINING SET DISCOVERY

Throughout this paper, we have assumed that the training set was partitioned into episodes by the teacher. Agents interacting with the world may not be given this explicit information. For example, a speech recognition system may not be told when it is conversing with a new speaker. Similarly, a character recognition system

would probably not be given explicit information about font changes. Learners can sometimes use the data itself to detect these changes, however. In many situations there is a strong prior that successive events are likely to have come from a single model with only occasional model changes. The EM algorithm is often used for segmenting unlabelled speech. It may be used in a similar manner to find the training set episode boundaries. First, a clustering algorithm is used to partition the training examples into episodes. A parameterized family is then fit to these episodes. The data is then repartitioned according to the similarity of the induced family parameters and the process is repeated until it converges. A similar approach may be applied when the model parameters vary slowly with time rather than occasionally jumping discontinuously.

## Acknowledgements

I'd like to thank Chris Bregler for work on the affine patch approach to surface learning, Alexander Linden for suggesting coupled maps for surface learning, and Peter Blicher for discussions.

## References

Baxter, J. (1995) Learning model bias. This volume.

Bregler, C. & Omohundro, S. (1994) Surface learning with applications to lipreading. In J. Cowan, G. Tesauro and J. Alspector (eds.), *Advances in Neural Information Processing Systems 6*, pp. 43-50. San Francisco, CA: Morgan Kaufmann Publishers.

Bregler, C. & Omohundro, S. (1995) Nonlinear image interpolation using manifold learning. In G. Tesauro, D. Touretzky and T. Leen (eds.), *Advances in Neural Information Processing Systems 7*. Cambridge, MA: MIT Press.

Bregler, C. & Omohundro, S. (1995) Nonlinear manifold learning for visual speech recognition. In W. Grimson (ed.), *Proceedings of the Fifth International Conference on Computer Vision*.

Jordan, M. & Jacobs, R. (1994) Hierarchical mixtures of experts and the EM algorithm. *Neural Computation*, 6:181-214.

MacKay, D. (1995) Probable networks and plausible predictions - a review of practical Bayesian methods for supervised neural networks. *Network*, to appear.

Pratt, L. (1994) Experiments on the transfer of knowledge between neural networks. In S. Hanson, G. Drastal, and R. Rivest (eds.), *Computational Learning Theory and Natural Learning Systems, Constraints and Prospects*, pp. 523-560. Cambridge, MA: MIT Press.

Revow, M., Williams, C. and Hinton, G. (1994) Using generative models for handwritten digit recognition. Technical report, University of Toronto.

Schmidhuber, J. (1995) On learning how to learn learning strategies. Technical Report FKI-198-94, Fakultät für Informatik, Technische Universität München.
